# Deep Representations and Codes for Image Auto-Annotation

**Ryan Kiros**
Department of Computing Science
University of Alberta
Edmonton, AB, Canada
rkiros@ualberta.ca

**Csaba Szepesvári**
Department of Computing Science
University of Alberta
Edmonton, AB, Canada
szepesva@ualberta.ca

## Abstract

The task of image auto-annotation, namely assigning a set of relevant tags to an image, is challenging due to the size and variability of tag vocabularies. Consequently, most existing algorithms focus on tag assignment and fix an often large number of hand-crafted features to describe image characteristics. In this paper we introduce a hierarchical model for learning representations of standard sized color images from the pixel level, removing the need for engineered feature representations and subsequent feature selection for annotation. We benchmark our model on the STL-10 recognition dataset, achieving state-of-the-art performance. When our features are combined with TagProp (Guillaumin et al.), we compete with or outperform existing annotation approaches that use over a dozen distinct handcrafted image descriptors. Furthermore, using 256-bit codes and Hamming distance for training TagProp, we exchange only a small reduction in performance for efficient storage and fast comparisons. Self-taught learning is used in all of our experiments and deeper architectures always outperform shallow ones.

## 1   Introduction

The development of successful methods for training deep architectures have influenced the development of representation learning algorithms either on top of SIFT descriptors [1, 2] or raw pixel input [3, 4, 5] for feature extraction of full-sized images. Algorithms for pixel-based representation learning avoid the use of any hand-crafted features, removing the difficulty of deciding which features are better suited for the desired task. Furthermore, self-taught learning [6] can be employed, taking advantage of feature learning from image databases independent of the target dataset.

Image auto-annotation is a multi-label classification task of assigning a set of relevant, descriptive tags to an image where tags often come from a vocabulary of hundreds to thousands of words. Figure 1 illustrates this task. Auto-annotation is a difficult problem due to the high variability of tags. Tags may describe objects, colors, scenes, local regions of the image (e.g. a building) or global characteristics (e.g. whether the image is outdoors). Consequently, many of the most successful annotation algorithms in the literature [7, 8, 9, 10, 11] have opted to focus on tag assignment and often fix a large number of hand-crafted features for input to their algorithms. The task of feature selection and applicability was studied by Zhang et al. [12] who utilized a group sparsity approach for dropping features. Furthermore, they observed that feature importance varied across datasets and some features led to redundancy, such as RGB and HSV histograms. Our main contribution in this paper is to remove the need to compute over a dozen hand-crafted features for annotating images and consequently remove the need for feature selection. We introduce a deep learning algorithm for learning hierarchical representations of full-sized color images from the pixel level, which may be seen as a generalization of the approach by Coates et al. [13] to larger images and more layers. We first benchmark our algorithm on the STL-10 recognition dataset, achieving a classification accuracy

of 62.1%. For annotation, we use the TagProp discriminitve metric learning algorithm [9] which has enjoyed state-of-the-art performance on popular annotation benchmarks. We test performance on three datasets: Natural scenes, IAPRTC-12 and ESP-Game. When our features are combined with TagProp, we either compete with or outperform existing methods when 15 distinct hand-crafted features and metrics are used. This gives the advantage of focusing new research on improving tag assignment algorithms without the need of deciding which features are best suited for the task.

| | | | | | |
|---|---|---|---|---|---|
|  | man<br>seat<br>train<br>woman | desk<br>hat<br>*man*<br>*train*<br>*woman* |  | balcony<br>car<br>church<br>front<br>house<br>people<br>side<br>street | building<br>lane<br>*people*<br>*street*<br>tower |  | jersey<br>short<br>sock<br>team<br>tree<br>uniform | *jersey*<br>*short*<br>*sock*<br>*team*<br>*tree* |
|  | mountain<br>road<br>snow<br>tree<br>white | hill<br>*road*<br>shadow<br>sky<br>*snow* |  | man<br>movie<br>people<br>show<br>*tv* | blue<br><br>tv |  | bike<br>dirt<br>jump<br>man<br>sky<br>tree | *bike*<br>blue<br>*jump*<br>*man*<br>*sky* |

Figure 1: Sample annotation results on IAPRTC-12 (top) and ESP-Game (bottom) using TagProp when each image is represented by a 256-bit code. The first column of tags is the gold standard and the second column are the predicted tags. Predicted tags that are italic are those that are also gold standard.

More recently, auto-annotation algorithms have focused on scalability to large databases with hundreds of thousands to millions of images. Such approaches include that of Tsai et al. [10] who construct visual synsets of images and Weston et al. [11] who used joint word-image embeddings. Our second contribution proposes the use of representing an image with a 256-bit code for annotation. Torralba et al. [14] performed an extensive analysis of small codes for image retrieval showing that even on databases with millions of images, linear search with Hamming distance can be performed efficiently. We utilize an autoencoder with a single hidden layer on top of our learned hierarchical representations to construct codes. Experimental results show only a small reduction in performance is obtained compared to the original learned features. In exchange, 256-bit codes are efficient to store and can be compared quickly with bitwise operations. To our knowledge, our approach is the first to learn binary codes from full-sized color images without the use of hand-crafted features. Such approaches often compute an initial descriptor such as GIST for representing an image. These approaches introduce too strong of a bottleneck too early, where the bottleneck in our pipeline comes after multiple layers of representation learning.

## 2 Hierarchical representation learning

In this section we describe our approach for learning a deep feature representation from the pixel-level of a color image. Our approach involves aspects of typical pipelines: pre-processing and whitening, dictionary learning, convolutional extraction and pooling. We define a module as a pass through each of the above operations. We first introduce our setup with high level descriptions followed by a more detailed descriptions of each stage. Finally, we show how to stack multiple modules on top of eachother.

Given a set of images, the learning phase of the network is as follows:

1. Extract randomly selected patches from each image and apply pre-processing.
2. Construct a dictionary using K-SVD.
3. Convolve the dictionary with larger tiles extracted across the image with a pre-defined stride length. Re-assemble the outputs in a non-overlapping, spatially preserving grid.
4. Pool over the reassembled features with a 2 layer pyramid.
5. Repeat the above operations for as many modules as desired.

For extracting features of a new image, we perform steps (3) and (4) for each module.

## 2.1 Patch extraction and pre-processing

Let $\{I^{(1)}, \ldots, I^{(m)}\}$ be a set of $m$ input images. For simplicity of explanation, assume $I^{(i)} \in \mathbb{R}^{n_V \times n_H \times 3}, i = 1 \ldots m$, though it need not be the case that all images are of the same size. Given a receptive field of size $r \times c$, we first extract $n_p$ patches across all images of size $r \times c \times 3$, followed by flatting each patch into a column vector. Let $X = \{x^{(1)}, \ldots, x^{(n_p)}\}, x^{(i)} \in \mathbb{R}^n, i = 1 \ldots n_p, n = 3rc$ denote the extracted patches. We first perform mean centering and unit variance scaling across features. This corresponds to local brightness and contrast normalization, respectively.

Next we follow [13] by performing ZCA whitening, which results in patches having zero mean, $\sum_{i=1}^{n_p} x^{(i)} = 0$, and identity covariance, $\frac{1}{n_p} \sum_{i=1}^{n_p} x^{(i)}(x^{(i)})^T = I$. A whitening matrix is computed as $W = V(Z + \epsilon I)^{-\frac{1}{2}} V^T$ where $C = VZV^T$ is an eigendecompostion of the centered covariance matrix $C = C(X)$ produced by mean subtraction of $M = M(X)$. The parameter $\epsilon$ is a small positive number having the effect of a low-pass filter.

## 2.2 Dictionary learning

Let $S = \{s^{(1)}, \ldots, s^{(n_p)}\}$ denote the whitened patches. We are now ready to construct a set of bases from $S$. We follow Bo et al. [5] and use K-SVD for learning a dictionary. K-SVD constructs a dictionary $D \in \mathbb{R}^{n \times k}$ and a sparse representation $\hat{S} \in \mathbb{R}^{k \times n_p}$ by solving the following optimization problem:

$$\underset{D, \hat{S}}{\text{minimize}} \quad \|S - D\hat{S}\|_F^2 \quad \text{subject to} \quad ||\hat{s}^{(i)}||_0 \leq q \quad \forall i \tag{1}$$

where $k$ is the desired number of bases. Optimization is done using alternation. When $D$ is fixed, the problem of obtaining $\hat{S}$ can be decomposed into $n_p$ subproblems of the form $\|s^{(i)} - D\hat{s}^{(i)}\|^2$ subject to $||\hat{s}^{(i)}||_0 \leq q \; \forall i$ which can be solved approximately using batch orthogonal matching pursuit [15]. When $\hat{S}$ is fixed, we update $D$ by first expressing equation 1 in terms of a residual $R^{(l)}$:

$$\|S - D\hat{S}\|_F^2 = \|S - \sum_{j \neq l} d^{(j)}\hat{s}^{(j)T} - d^{(l)}\hat{s}^{(l)T}\|_F^2 = \|R^{(l)} - d^{(l)}\hat{s}^{(l)T}\|_F^2 \tag{2}$$

where $l \in \{1, \ldots, k\}$. A solution for $d^{(l)}$, the $l$-th column of $D$, can be obtained through an SVD of $R^{(l)}$. For space considerations, we refer the reader to Rubinstein et al. [15] for more details. [1]

## 2.3 Convolutional feature extraction

Given an image $I^{(i)}$, we first partition the image into a set of tiles $T^{(i)}$ of size $n_t \times n_t$ with a predefined stride length $s$ between each tile. Each patch in tile $T_t^{(i)}$ is processed in the same way as before dictionary construction (mean centering, contrast normalization, whitening) for which the mean and whitening matrices $M$ and $W$ are used. Let $T_{tj}^{(i)}$ denote the $t$-th tile and $j$-th channel with respect to image $I^{(i)}$ and let $D_j^{(l)} \in \mathbb{R}^{r \times c}$ denote the $l$-th basis for channel $j$ of $D$. The encoding $f_{tl}^{(i)}$ for tile $t$ and basis $l$ is given by:

$$f_{tl}^{(i)} = \max\left\{\tanh\left(\sum_{j=1}^{3} T_{tj}^{(i)} * D_j^{(l)}\right), 0\right\} \tag{3}$$

where * denotes convolution and max and tanh operations are applied componentwise. Even though it is not the associated encoding with K-SVD, this type of 'surrogate coding' was studied by Coates

et al. [13]. Let $f_t^{(i)}$ denote the concatenated encodings over bases, which have a resulting dimension of $(n_t - r + 1) \times (n_t - c + 1) \times k$. These are then re-assembled into spatial-preserving, non-overlapping regions. See figure 2 for an illustration. We perform one additional localized contrast normalization over $f_t^{(i)}$ of the form $f_t^{(i)} \leftarrow (f_t^{(i)} - \mu(f_t^{(i)}))/\max\{\mu(\sigma_t), \sigma_t^{(i)}\}$. Similar types of normalization have been shown to be critical for performance by Ranzato et al. [16] and Bo et al. [5].

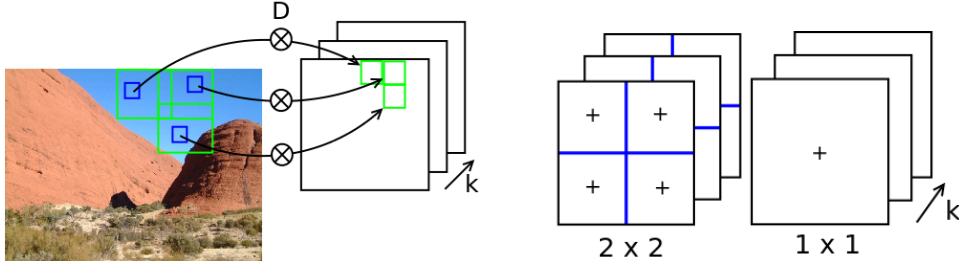

Figure 2: Left: $D$ is convolved with each tile (large green square) with receptive field (small blue square) over a given stride. The outputs are re-assembled in non-overlapping regions preserving spatial structure. Right: $2 \times 2$ and $1 \times 1$ regions are summed (pooled) along each cross section.

## 2.4 Pooling

The final step of our pipeline is to perform spatial pooling over the re-assembled regions of the encodings $f_t^{(i)}$. Consider the $l$-th cross section corresponding to the $l$-th dictionary element, $l \in \{1, \ldots, k\}$. We may then pool over each of the spatial regions of this cross section by summing over the activations of the corresponding spatial regions. This is done in the form of a 2-layer spatial pyramid, where the base of the pyramid consists of 4 blocks of $2 \times 2$ tiling and the top of the pyramid consisting of a single block across the whole cross section. See figure 2 for an illustration.

Once pooling is performed, the re-assembled encodings result in a shape of size $1 \times 1 \times k$ and $2 \times 2 \times k$ from each layer of the pyramid. To obtain the final feature vector, each layer is flattened into a vector and the resulting vectors are concatinated into a single long feature vector of dimension $5k$ for each image $I^{(i)}$. Prior to classification, these features are normalized to have zero mean and unit variance.

## 2.5 Training multiple modules

What we have described up until now is how to extract features using a single module corresponding to dictionary learning, extraction and pooling. We can now extend this framework into a deep network by stacking multiple modules. Once the first module has been trained, we can take the pooled features to be input to a second module. Freezing the learned dictionary from the first module, we can then apply all the same steps a second time to the pooled representations. This type of stacked training can be performed to as many modules as desired.

To be more specific on the input to the second module, we use an additional spatial pooling operation on the re-assembled encodings of the first module, where we extract 256 blocks of $16 \times 16$ tiling, resulting in a representation of size $16 \times 16 \times k$. It is these inputs which we then pass on to the second module. We choose to use $16 \times 16$ as a trade off to aggregating too much information and increasing memory and time complexity. As an illustration, the same operations for the second module are used as in figure 2 except the image is replaced with the $16 \times 16 \times k$ pooled features. In the next module, the number of channels is equal to the number of bases from the previous module.

# 3   Code construction and discriminitive metric learning

In this section we first show to learn binary codes from our learned features, followed by a review of the TagProp algorithm [9] used for annotation.

## 3.1 Learning binary codes for annotation

Our codes are learned by adding an autoencoder with a single hidden layer on top of the learned output representations. Let $f^{(i)} \in \mathbb{R}^{d_m}$ denote the learned representation for image $I^{(i)}$ of dimension $d_m$ using either a one or two module architecture. The code $b^{(i)}$ for $f^{(i)}$ is computed by $b^{(i)} = \text{round}(\sigma(f^{(i)}))$ where $\sigma(f^{(i)}) = (1 + \exp(Wf^{(i)} + \beta))^{-1}$, $W \in \mathbb{R}^{d_b \times d_m}$, $\beta \in \mathbb{R}^{d_b}$ and $d_b$ is the number of bits (in our case, $d_b = 256$). Using a linear output layer, our objective is to minimize the mean squared error of reconstructions of the the inputs given by $\frac{1}{m} \sum_i \left[ (\tilde{W}\sigma(f^{(i)}) + \tilde{\beta}) - f^{(i)} \right]^2$, where $\tilde{W} \in \mathbb{R}^{d_m \times d_b}$, $\tilde{\beta} \in \mathbb{R}^{d_m}$ are the second layer weights and biases respectively. The objective is minimized using standard backpropagation.

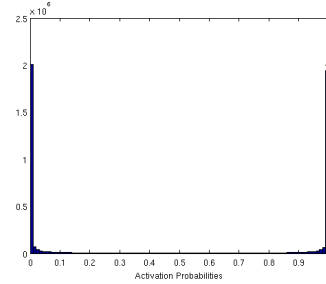

Figure 3: Coding layer activation values after training the autoencoder.

As is, the optimization does not take into consideration the rounding used in the coding layer and consequently the output is not adapted for this operation. We follow Salakhutdinov et al. [17] and use additive 'deterministic' Gaussian noise with zero mean in the coding layer that is fixed in advance for each datapoint when performing a bottom-up pass through the network. Using unit variance was sufficient to force almost all the activations near $\{0, 1\}$. We tried other approaches, including simple thresholding but found the Gaussian noise to be most successful without interfering with the optimization. Figure 3 shows the coding layer activation values after backpropagation when noise has been added.

## 3.2 The tag propagation (TagProp) algorithm

Let $V$ denote a fixed vocabulary of tags and $I$ denote a list of input images. Our goal at test time, given a new input image $i'$, is to assign a set of tags $v \in V$ that are most relevant to the content of $i'$. TagProp operates on pairwise distances to learn a conditional distribution of words given images. More specifically, let $y_{iw} \in \{1, -1\}, i \in I, w \in V$ be an indicator for whether tag $w$ is present in image $i$. In TagProp, the probability that $y_{iw} = 1$ is given by $\sigma(\alpha_w x_{iw} + \beta_w), x_{iw} = \sum_j \pi_{ij} y_{jw}$ where $\sigma(z) = (1 + \exp(-z))^{-1}$ is the logistic function, $(\alpha_w, \beta_w)$ are word-specific model parameters to be estimated and $\pi_{ij}$ are distance-based weights also to be estimated. More specifically, $\pi_{ij}$ is expressed as

$$\pi_{ij} = \frac{\exp(-d_h(i,j))}{\sum_{j'} \exp(-d_h(i,j'))}, \quad d_h(i,j) = hd_{ij}, \quad h \geq 0 \tag{4}$$

where we shall call $d_{ij}$ the *base* distance between images $i$ and $j$. Let $\theta = \{\alpha_w \forall w \in V, \beta_w \forall w \in V, h\}$ denote the list of model parameters. The model is trained to maximize the following quasi-likelihood of the data given by $\mathcal{L} = \sum_{i,w} c_{iw} \log p(y_{iw}), c_{iw} = \frac{1}{n_+}$ if $y_{iw} = 1$ and $\frac{1}{n_-}$ otherwise, where $n_+$ is the total number of positive labels of $w$ and likewise for $n_-$ and missing labels. This weighting allows us to take into account imbalances between label presence and absence. Combined with the logistic word models, it accounts for much higher recall in rare tags which would normally be less likely to be recalled in a basic $k$-NN setup. Optimization of $\mathcal{L}$ is performed using a projected gradient method for enforcing the non-negativity constraint in $h$.

The choice of base distance used depends on the image representation. In the above description, the model was derived assuming only a single base distance is computed between images. This can be generalized to an arbitrary number of distances by letting $h$ be a parameter vector and letting $d_h(i,j)$ be a weighted combination of distances in $h$. Under this formulation, multiple descriptors of images can be computed and weighted. The best performance of TagProp [9] was indeed obtained using this multiple metric formulation in combination with the logistic word models. In our case, Euclidean distance is used and Hamming distance for binary codes. Furthermore, we only consider pairwise distances from the $K$ nearest neighbors, where $K$ is chosen though cross validation.

## 4 Experiments

We perform experimental evaluation of our methods on 4 datasets: one dataset, STL-10, for object recognition to benchmark our hierarchical model and three datasets for annotation: Natural Scenes, IAPRTC-12 and ESP-Game [2] .

For all our experiments, we use $k_1 = 512$ first module bases, $k_2 = 1024$ second module bases, receptive field sizes of $6 \times 6$ and $2 \times 2$ and tile sizes $(n_t)$ of $16 \times 16$ and $6 \times 6$. The total number of features for the combined first and second module representation is thus $5(k_1 + k_2) = 7680$. Images are resized such that the longest side is no larger than 300 pixels with preserved aspect ratio. The first module stride length is chosen based on the length of the longest side of the image: 4 if the side is less than 128 pixels, 6 if less than 214 pixels and 8 otherwise. The second module stride length is fixed at 2. For training the autoencoder, we use 10 epochs (passes over the training set) with minibatches of size no larger than 1000. Optimization is done using Polak Ribiere conjugate gradients with 3 linesearches per minibatch. [3]

We also incorporate the use of self-taught learning [6] in our annotation experiments by utilizing the Mirflickr dataset for dictionary learning. Mirflickr is a collection of 25000 images taken from flickr and deemed to have high interestness rating. We randomly sampled 10000 images from this dataset for training K-SVD on both modules. All reported results for Natural Scenes, IAPRTC-12 and ESP-Game use self-taught learning. Our code for feature learning will be made available online.

### 4.1 STL-10

The STL-10 dataset is a collection of $96 \times 96$ images of 10 classes with images partitioned into 10 folds of 1000 images each and a test set of size 8000. Alongside these labeled images is a set of 100000 unlabeled images that may or may not come from the same distribution as the training data. The evaluation procedure is to perform representation learning on the unlabeled data and apply the representations to the training set, averaging test errors across all folds. We randomly chose 10000 images from the unlabeled set for training and use a linear L2-SVM for classification with 5-fold cross validation for model selection.

Table 1: A selection of the best results obtained on the STL-10 dataset.

| Method | Accuracy |
|---|---|
| Sparse filtering [18] | 53.5% |
| OMP, $k = 1600$ [13] | 54.9% |
| OMP, SC encoder, $k = 1600$ [13] | 59.0% |
| Receptive field learning, 3 modules [19] | 60.1% |
| Video unsup features [20] | 61.0% |
| Hierarchical matching persuit [21] | 64.5% |
| 1st Module | 56.4 % |
| 1st + 2nd Module | 62.1 % |

Table 1 shows our results on STL-10. Our 2 module architecture outperforms all existing approaches except for the recently proposed hierarchical matching pursuit (HMP). HMP uses joint layerwise pooling and separate training for RGB and grayscale dictionaries, approaches which may also be adapted to our method. Moreover, we hypothesize that further improvements can be made when the receptive field learning strategies of Coates et al. [19] and Jia et al. [22] are incorporated into a third module.

### 4.2 Natural scenes

The Natural Scenes dataset is a multi-label collection of 2000 images from 5 classes: desert, forest, mountain, ocean and sunset. We follow standard protocol and report the average results of 5 metrics using 10 fold cross validation: Hamming loss (**HL**), one error (**OE**), coverage (**C**), ranking loss (**RL**) and average precision (**AP**). For space considerations, these metrics are defined in the appendix. To perform model selection with TagProp, we perform 5-fold cross validation with each of the 10-folds to determine the value of $K$ which minimizes Hamming loss.

Table 2: A selection of the best results obtained on the Natural Scenes dataset. Arrows indicate direction of improved performance.

| Method | HL ↓ | OE ↓ | C ↓ | RL ↓ | AP ↑ |
|---|---|---|---|---|---|
| ML-KNN [23] | 0.169 | 0.300 | 0.939 | 0.168 | 0.803 |
| ML-I2C [24] | 0.159 | 0.311 | 0.883 | 0.156 | 0.804 |
| InsDif [25] | 0.152 | 0.259 | 0.834 | 0.140 | 0.830 |
| ML-LI2C [24] | 0.129 | 0.190 | 0.624 | 0.091 | 0.881 |
| 1st Module | 0.113 | 0.170 | 0.580 | 0.080 | 0.895 |
| 1st Module, 256-bit | 0.113 | 0.169 | 0.585 | 0.082 | 0.894 |
| 1st + 2nd Module | 0.100 | 0.140 | 0.554 | 0.074 | 0.910 |
| 1st + 2nd Module, 256-bit | 0.106 | 0.155 | 0.558 | 0.075 | 0.903 |

Table 3: A selection of the best results obtained on the IAPRTC-12 dataset (left) and ESP-Game (right) datasets.

| Method | P | R | N+ | P | R | N+ |
|---|---|---|---|---|---|---|
| MBRM [26] | 0.24 | 0.23 | 223 | 0.18 | 0.19 | 209 |
| LASSO [7] | 0.28 | 0.29 | 246 | 0.21 | 0.24 | 224 |
| JEC [7] | 0.28 | 0.29 | 250 | 0.22 | 0.25 | 224 |
| GS [12] | 0.32 | 0.29 | 252 | - | - | - |
| CCD [8] | 0.44 | 0.29 | 251 | 0.36 | 0.24 | 232 |
| TagProp ($\sigma$ SD) [9] | 0.41 | 0.30 | 259 | 0.39 | 0.24 | 232 |
| TagProp ($\sigma$ ML) [9] | 0.46 | 0.35 | 266 | 0.39 | 0.27 | 239 |
| 1st Module | 0.37 | 0.25 | 241 | 0.37 | 0.20 | 231 |
| 1st Module, 256-bit | 0.34 | 0.22 | 236 | 0.35 | 0.20 | 231 |
| 1st + 2nd Module | 0.42 | 0.29 | 252 | 0.38 | 0.22 | 228 |
| 1st + 2nd Module, 256-bit | 0.36 | 0.25 | 244 | 0.37 | 0.23 | 236 |

Table 2 shows the results of our method. In all five measures we obtain improvement over previous methods. Furthermore, using 256-bit codes offers near equivalent performance. As in the case of STL-10, improvements are made over a single module.

## 4.3 IAPRTC-12 and ESP-Game

IAPRTC-12 is a collection of 20000 images with a vocabulary size of $|V| = 291$ and an average of 5.7 tags per image. ESP-Game is a collection of 60000 images with $|V| = 268$ and an average of 4.7 tags per class. Following Guillaumin et al. [9] we apply experiments to a pre-defined subset of 20000 images. Using standard protocol performance is evaluated using 3 measures: precision (**P**), recall (**R**) and the number of recalled tags (**N+**). $N+$ indicates the number of tags that were recalled at least once for annotation on the test set. Annotations are made by choosing the 5 most probable tags for each image as is done with previous evaluations. As with the natural scenes dataset, we perform 5-fold cross validation to determine $K$ for training TagProp.

Table 3 shows our results with IAPRTC-12 on the left and ESP-Game on the right. Our results give comparable performance to CCD and the single distance (SD) variation of TagProp. Unfortunately, we are unable to match the recall values obtained with the multiple metric (ML) variation of TagProp. Of importance, we outperform GS who specifically studied the use of feature selection. Our 256-bit codes suffer a loss of performance on IAPRTC-12 but give near equivalent results on ESP-Game. We note again that our features were learned on an entirely different dataset (Mirflickr) in order to show their generalization capabilities.

Finally, we perform two qualitative experiments. Figure 4 shows sample unsupervised retrieval results using the learned 256-bit codes on IAPRTC-12 and ESP-Game while figure 5 illustrates sample annotation performance when training on one dataset and annotating the other. These results show that our codes are able to capture high-level semantic concepts that perform well for retrieval and transfer learning across datasets. We note however, that when annotating ESP-game when training was done on IAPRTC-12 led to more false human annotations (such as the bottom-right

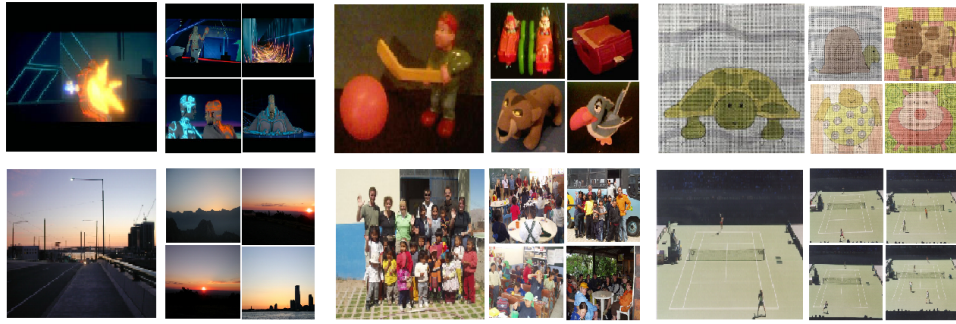

Figure 4: Sample 256-bit unsupervised retrieval results on ESP-Game (top) and IAPRTC-12 (bottom). A query image from the test set is used to retrieve the four nearest neighbors from the training set.

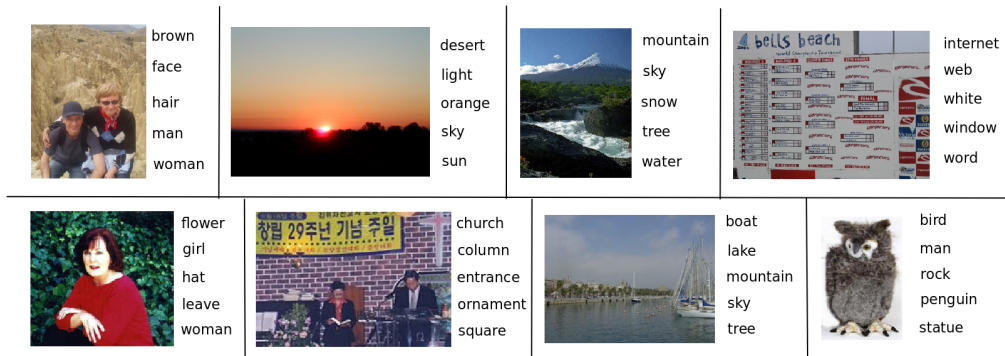

Figure 5: Sample 256-bit annotation results when training on one dataset and annotating the other. Top: Training on ESP-Game, annotation on IAPRTC-12. Bottom: Training on IAPRTC-12, annotation on ESP-Game.

image in figure 5). We hypothesize that this is due to a larger proportion of persons in the IAPRTC-12 training set.

# 5    Conclusion

In this paper we introduced a hierarchical model for learning feature representations of standard sized color images for the task of image annotation. Our results compare favorably to existing approaches that use over a dozen handcrafted image descriptors.

Our primary goal for future work is to test the effectiveness of this approach on web-scale annotation systems with millions of images. The success of self-taught learning in this setting means only one dictionary per module ever needs to be learned. Furthermore, our features can be used in combination with any nearest neighbor based algorithm for annotation. It is our hope that the successful use of binary codes for annotation will allow further research to bridge the gap between the annotation algorithms used on small scale problems to those required for web scale tasks. We also intend to evaluate the effectiveness of semantic hashing on large databases when much smaller codes are used. Krizhevsky et al. [27] evaluated semantic hashing using very deep autoencoders on tiny ($32 \times 32$) images. Future work also involves performing similar experiments but on standard sized RGB images.

### Acknowledgments

The authors thank Axel Soto as well as the anonymous reviewers for helpful discussion and comments. This work was funded by NSERC and the Alberta Innovates Centre for Machine Learning.

## Footnotes

[1]We use Rubinstein's implementation available at `http://www.cs.technion.ac.il/~ronrubin/software.html`

[2]Tags for IAPRTC-12 and ESP-Game as well as the features used by existing approaches can be found at `http://lear.inrialpes.fr/people/guillaumin/data.php`

[3]Rasmussen's minimize routine is used.

# References

[1] T Huang. Linear spatial pyramid matching using sparse coding for image classification. *CVPR*, pages 1794–1801, 2009.

[2] K. Yu F. Lv T. Huang J. Wang, J. Yang and Y. Gong. Locality-constrained linear coding for image classification. In *CVPR*, pages 3360–3367, 2010.

[3] R. Ranganath H Lee, R. Grosse and A.Y. Ng. Convolutional deep belief networks for scalable unsupervised learning of hierarchical representations. *ICML*, pages 1–8, 2009.

[4] K. Yu, Y. Lin, and J. Lafferty. Learning image representations from the pixel level via hierarchical sparse coding. In *CVPR*, pages 1713–1720, 2011.

[5] L. Bo, X. Ren, and D. Fox. Hierarchical Matching Pursuit for Image Classification: Architecture and Fast Algorithms. In *NIPS*, 2011.

[6] Rajat Raina, Alexis Battle, Honglak Lee, Benjamin Packer, and Andrew Y Ng. *Self-taught learning*, pages 759–766. ICML, 2007.

[7] A. Makadia, V. Pavlovic, and S. Kumar. A new baseline for image annotation. In *ECCV*, volume 8, pages 316–329, 2008.

[8] H. Nakayama. *Linear Distance Metric Learning for Large-scale Generic Image Recognition*. PhD thesis, The University of Tokyo.

[9] M. Guillaumin, T. Mensink, J. Verbeek, and C. Schmid. Tagprop: Discriminative metric learning in nearest neighbor models for image auto-annotation. In *ICCV*, pages 309–316, 2009.

[10] D. Tsai, Y. Jing, Y. Liu, H.A. Rowley, S. Ioffe, and J.M. Rehg. Large-scale image annotation using visual synset. In *ICCV*, pages 611–618, 2011.

[11] J. Weston, S. Bengio, and N. Usunier. Large scale image annotation: learning to rank with joint word-image embeddings. *Machine Learning*, 81(1):21–35, 2010.

[12] S. Zhang, J. Huang, Y. Huang, Y. Yu, H. Li, and D.N. Metaxas. Automatic image annotation using group sparsity. In *CVPR*, pages 3312–3319, 2010.

[13] A. Coates and A.Y. Ng. The importance of encoding versus training with sparse coding and vector quantization. In *ICML*, 2011.

[14] A. Torralba, R. Fergus, and Y. Weiss. Small codes and large image databases for recognition. In *CVPR*, pages 1–8, 2008.

[15] R. Rubinstein, M. Zibulevsky, and M. Elad. Efficient implementation of the $k$-SVD algorithm using batch orthogonal matching pursuit. *Technical Report*, 2008.

[16] M. Ranzato K. Jarrett, K. Kavukcuoglu and Y. LeCun. What is the best multi-stage architecture for object recognition? *ICCV*, 6:2146–2153, 2009.

[17] G. Hinton and R. Salakhutdinov. Discovering binary codes for documents by learning deep generative models. *Topics in Cognitive Science*, 3(1):74–91, 2011.

[18] Z. Chen S. Bhaskar J. Ngiam, P. W. Koh and A.Y. Ng. Sparse filtering. *NIPS*, 2011.

[19] A. Coates and A.Y. Ng. Selecting receptive fields in deep networks. *NIPS*, 2011.

[20] W. Zou, A. Ng, and Kai. Yu. Unsupervised learning of visual invariance with temporal coherence. In *NIPS Workshop on Deep Learning and Unsupervised Feature Learning*, 2011.

[21] L. Bo, X. Ren, and D. Fox. Unsupervised Feature Learning for RGB-D Based Object Recognition. In *ISER*, June 2012.

[22] Y. Jia, C. Huang, and T. Darrell. Beyond spatial pyramids: Receptive field learning for pooled image features. In *CVPR*, 2012.

[23] M.L. Zhang and Z.H. Zhou. ML-KNN: A lazy learning approach to multi-label learning. *Pattern Recognition*, 40(7):2038–2048, 2007.

[24] Y. Hu Z. Wang and L.T. Chia. Multi-label learning by image-to-class distance for scene classification and image annotation. In *CIVR*, pages 105–112, 2010.

[25] M.L. Zhang and Z.H. Zhou. Multi-label learning by instance differentiation. In *AAAI*, number 1, pages 669–674, 2007.

[26] SL Feng, R. Manmatha, and V. Lavrenko. Multiple Bernoulli relevance models for image and video annotation. In *CVPR*, pages 1002–1009, 2004.

[27] A. Krizhevsky and G.E. Hinton. Using very deep autoencoders for content-based image retrieval. ESANN, 2011.

